# Emergence of Object-Selective Features in Unsupervised Feature Learning

**Adam Coates, Andrej Karpathy, Andrew Y. Ng**
Computer Science Department
Stanford University
Stanford, CA 94305
{acoates,karpathy,ang}@cs.stanford.edu

## Abstract

Recent work in unsupervised feature learning has focused on the goal of discovering high-level features from unlabeled images. Much progress has been made in this direction, but in most cases it is still standard to use a large amount of labeled data in order to construct detectors sensitive to object classes or other complex patterns in the data. In this paper, we aim to test the hypothesis that unsupervised feature learning methods, provided with only unlabeled data, can learn high-level, invariant features that are sensitive to commonly-occurring objects. Though a handful of prior results suggest that this is possible when each object class accounts for a large fraction of the data (as in many labeled datasets), it is unclear whether something similar can be accomplished when dealing with completely unlabeled data. A major obstacle to this test, however, is scale: we cannot expect to succeed with small datasets or with small numbers of learned features. Here, we propose a large-scale feature learning system that enables us to carry out this experiment, learning 150,000 features from tens of millions of unlabeled images. Based on two scalable clustering algorithms (K-means and agglomerative clustering), we find that our simple system can discover features sensitive to a commonly occurring object class (human faces) and can also combine these into detectors invariant to significant global distortions like large translations and scale.

## 1   Introduction

Many algorithms are now available to learn hierarchical features from unlabeled image data. There is some evidence that these algorithms are able to learn useful high-level features without labels, yet in practice it is still common to train such features from labeled datasets (but ignoring the labels), and to ultimately use a supervised learning algorithm to learn to detect more complex patterns that the unsupervised learning algorithm is unable to find on its own. Thus, an interesting open question is whether unsupervised feature learning algorithms are able to construct features, without the benefit of supervision, that can identify high-level concepts like frequently-occurring object classes. It is already known that this can be achieved when the dataset is sufficiently restricted that object classes are clearly defined (typically closely cropped images) and occur very frequently [13, 21, 22]. In this work we aim to test whether unsupervised learning algorithms can achieve a similar result without any supervision at all.

The setting we consider is a challenging one. We have harvested a dataset of 1.4 million image thumbnails from YouTube and extracted roughly 57 million 32-by-32 pixel patches at random locations and scales. These patches are very different from those found in labeled datasets like CIFAR-10 [9]. The overwhelming majority of patches in our dataset appear to be random clutter. In the cases where such a patch contains an identifiable object, it may well be scaled, arbitrarily cropped, or uncentered. As a result, it is very unclear where an "object class" begins or ends in this type of patch dataset, and less clear that a completely unsupervised learning algorithm could manage to cre-

ate "object-selective" features able to distinguish an object from the wide variety of clutter without some other type of supervision.

In order to have some hope of success, we can identify several key properties that our learning algorithm should likely have. First, since identifiable objects show up very rarely, it is clear that we are obliged to train from extremely large datasets. We have no way of controlling how often a particular object shows up and thus enough data must be used to ensure that an object class is seen many times—often enough that it cannot be disregarded as random clutter. Second, we are also likely to need a very large number of features. Training too few features will cause us to "under-fit" the distribution, forcing the learning algorithm to ignore rare events like objects. Finally, as is already common in feature learning work, we should aim to build features that incorporate invariance so that features respond not just to a specific pattern (e.g., an object at a single location and scale), but to a range of patterns that collectively belong to the same object class (e.g., the same object seen at many locations and scales). Unfortunately, these desiderata are difficult to achieve at once: current methods for building invariant hierarchies of features are difficult to scale up to train many thousands of features from our 57 million patch dataset on our cluster of 30 machines.

In this paper, we will propose a highly scalable combination of clustering algorithms for learning selective and invariant features that are capable of tackling this size of problem. Surprisingly, we find that despite the simplicity of these algorithms we are nevertheless able to discover high-level features sensitive to the most commonly occurring object class present in our dataset: human faces. In fact, we find that these features are better face detectors than a linear filter trained from labeled data, achieving up to 86% AUC compared to 77% on labeled validation data. Thus, our results emphasize that not only can unsupervised learning algorithms discover object-selective features with no labeled data, but that such features can potentially perform better than basic supervised detectors due to their deep architecture. Though our approach is based on fast clustering algorithms (K-means and agglomerative clustering), its basic behavior is essentially similar to existing methods for building invariant feature hierarchies, suggesting that other popular feature learning methods currently available may also be able to achieve such results if run at large enough scale. Indeed, recent work with a more sophisticated (but vastly more expensive) feature-learning algorithm appears to achieve similar results [11] when presented with full-frame images.

We will begin with a description of our algorithms for learning selective and invariant features, and explain their relationship to existing systems. We will then move on to presenting our experimental results. Related results and methods to our own will be reviewed briefly before concluding.

## 2 Algorithm

Our system is built on two separate learning modules: (i) an algorithm to learn selective features (linear filters that respond to a specific input pattern), and (ii) an algorithm to combine the selective features into invariant features (that respond to a spectrum of gradually changing patterns). We will refer to these features as "simple cells" and "complex cells" respectively, in analogy to previous work and to biological cells with (very loosely) related response properties. Following other popular systems [14, 12, 6, 5] we will then use these two algorithms to build alternating layers of simple cell and complex cell features.

### 2.1 Learning Selective Features (Simple Cells)

The first module in our learning system trains a bank of linear filters to represent our selective "simple cell" features. For this purpose we use the K-means-like method used by [2], which has previously been used for large-scale feature learning.

The algorithm is given a set of input vectors $x^{(i)} \in \Re^n, i = 1, \ldots, m$. These vectors are pre-processed by removing the mean and normalizing each example, then performing PCA whitening. We then learn a dictionary $D \in \Re^{n \times d}$ of linear filters as in [2] by alternating optimization over filters $D$ and "cluster assignments" $C$:

$$\underset{D,C}{\text{minimize}} \, ||DC^{(i)} - x^{(i)}||_2^2$$

$$\text{subject to } ||D^{(j)}||_2 = 1, \forall j,$$

$$\text{and } ||C^{(i)}||_0 \leq 1, \forall i.$$

Here the constraint $||C^{(i)}||_0 \leq 1$ means that the vectors $C^{(i)}, i = 1, \ldots, m$ are allowed to contain only a single non-zero, but the non-zero value is otherwise unconstrained. Given the linear filters $D$, we then define the responses of the learned simple cell features as $s^{(i)} = g(a^{(i)})$ where $a^{(i)} = D^\top x^{(i)}$ and $g(\cdot)$ is a nonlinear activation function. In our experiments we will typically use $g(a) = |a|$ for the first layer of simple cells, and $g(a) = a$ for the second.[1]

## 2.2 Learning Invariant Features (Complex Cells)

To construct invariant complex cell features a common approach is to create "pooling units" that combine the responses of lower-level simple cells. In this work, we use max-pooling units [14, 13]. Specifically, given a vector of simple cell responses $s^{(i)}$, we will train complex cell features whose responses are given by:

$$c_j^{(i)} = \max_{k \in G_j} s_k^{(i)}$$

where $G_j$ is a set that specifies which simple cells the $j$'th complex cell should pool over. Thus, the complex cell $c_j$ is an invariant feature that responds significantly to any of the patterns represented by simple cells in its group.

Each group $G_j$ should specify a set of simple cells that are, in some sense, similar to one another. In convolutional neural networks [12], for instance, each group is hard-coded to include translated copies of the same filter resulting in complex cell responses $c_j$ that are invariant to small translations. Some algorithms [6, 3] fix the groups $G_j$ ahead of time then optimize the simple cell filters $D$ so that the simple cells in each group share a particular form of statistical dependence. In our system, we will use linear correlation of simple cell responses as our similarity metric, $\mathbb{E}[a_k a_l]$, and construct groups $G_j$ that combine similar features according to this metric. Computing the similarity directly would normally require us to estimate the correlations from data, but since the inputs $x^{(i)}$ are whitened we can instead compute the similarity directly from the filter weights:

$$\mathbb{E}[a_k a_l] = \mathbb{E}[D^{(k)\top} x^{(i)} x^{(i)\top} D^{(l)}] = D^{(k)\top} D^{(l)}.$$

For convenience in the following, we will actually use the dissimilarity between features, defined as $d(k, l) = ||D^{(k)} - D^{(l)}||_2 = \sqrt{2 - 2\mathbb{E}[a_k a_l]}$.

To construct the groups $G$, we will use a version of single-link agglomerative clustering to combine sets of features that have low dissimilarity according to $d(k, l)$.[2] To construct a single group $G_0$ we begin by choosing a random simple cell filter, say $D^{(k)}$, as the first member. We then search for candidate cells to be added to the group by computing $d(k, l)$ for each simple cell filter $D^{(l)}$ and add $D^{(l)}$ to the group if $d(k, l)$ is less than some limit $\tau$. The algorithm then continues to expand $G_0$ by adding any additional simple cells that are closer than $\tau$ to any one of the simple cells already in the group. This procedure continues until there are no more cells to be added, or until the diameter of the group (the dissimilarity between the two furthest cells in the group) reaches a limit $\Delta$.[3]

This procedure can be executed, quite rapidly, in parallel for a large number of randomly chosen simple cells to act as the "seed" cell, thus allowing us to train many complex cells at once. Compared to the simple cell learning procedure, the computational cost is extremely small even for our rudimentary implementation. In practice, we often generate many groups (e.g., several thousand) and then keep only a random subset of the largest groups. This ensures that we do not end up with many groups that pool over very few simple cells (and hence yield complex cells $c_j$ that are not especially invariant).

## 2.3 Algorithm Behavior

Though it seems plausible that pooling simple cells with similar-looking filters according to $d(k, l)$ as above should give us some form of invariant feature, it may not yet be clear why this form of

invariance is desirable. To explain, we will consider a simple "toy" data distribution where the behavior of these algorithms is more clear. Specifically, we will generate three heavy-tailed random variables $X, Y, Z$ according to:

$$\sigma_1, \sigma_2 \sim \mathcal{L}(0, \lambda)$$
$$e_1, e_2, e_3 \sim \mathcal{N}(0, 1)$$
$$X = e_1 \sigma_1, Y = e_2 \sigma_1, Z = e_3 \sigma_2$$

Here, $\sigma_1, \sigma_2$ are scale parameters sampled independently from a Laplace distribution, and $e_1, e_2, e_3$ are sampled independently from a unit Gaussian. The result is that $Z$ is independent of both $X$ and $Y$, but $X$ and $Y$ are not independent due to their shared scale parameter $\sigma_1$ [6]. An isocontour of the density of this distribution is shown in Figure 1a.

Other popular algorithms [6, 5, 3] for learning complex-cell features are designed to identify $X$ and $Y$ as features to be pooled together due to the correlation in their energies (scales). One empirical motivation for this kind of invariance comes from natural images: if we have three simple-cell filter responses $a_1 = D^{(1)\top}x$, $a_2 = D^{(2)\top}x$, $a_3 = D^{(3)\top}x$ where $D^{(1)}$ and $D^{(2)}$ are Gabor filters in quadrature phase, but $D^{(3)}$ is a Gabor filter at a different orientation, then the responses $a_1, a_2, a_3$ will tend to have a distribution very similar to the model of $X, Y, Z$ above [7]. By pooling together the responses of $a_1$ and $a_2$ a complex cell is able to detect an edge of fixed orientation invariant to small translations. This model also makes sense for higher-level invariances where $X$ and $Y$ do not merely represent responses of linear filters on image patches but feature responses in a deep network. Indeed, the $X$–$Y$ plane in Figure 1a is referred to as an "invariant subspace" [8].

Our combination of simple cell and complex cell learning algorithms above tend to learn this same type of invariance. After whitening and normalization, the data points $X, Y, Z$ drawn from the distribution above will lie (roughly) on a sphere. The density of these data points is pictured in Figure 1b, where it can be seen that the highest density areas are in a "belt" in the $X$–$Y$ plane and at the poles along the $Z$ axis with a low-density region in between. Application of our K-means clustering method to this data results in centroids shown as $*$ marks in Figure 1b. From this picture it is clear what a subsequent application of our single-link clustering algorithm will do: it will try to string together the centroids around the "belt" that forms the invariant subspace and avoid connecting them to the (distant) centroids at the poles. Max-pooling over the responses of these filters will result in a complex cell that responds consistently to points in the $X$–$Y$ plane, but not in the $Z$ direction— that is, we end up with an invariant feature detector very similar to those constructed by existing methods. Figure 1c depicts this result, along with visualizations of the hypothetical gabor filters $D^{(1)}, D^{(2)}, D^{(3)}$ described above that might correspond to the learned centroids.

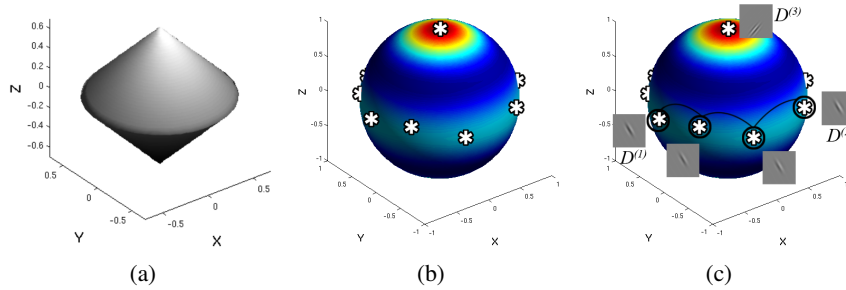

(a)          (b)          (c)

Figure 1: (a) An isocontour of a sparse probability distribution over variables X, Y, and Z. (See text for details.) (b) A visualization of the spherical density obtained from the distribution in (a) after normalization. Red areas are high density and dark blue areas are low density. Centroids learned by K-means from this data are shown on the surface of the sphere as * marks. (c) A pooling unit identified by applying single-link clustering to the centroids (black links join pooled filters). (See text.)

## 2.4 Feature Hierarchy

Now that we have defined our simple and complex cell learning algorithms, we can use them to train alternating layers of selective and invariant features. We will train 4 layers total, 2 of each type. The architecture we use is pictured in Figure 2a.

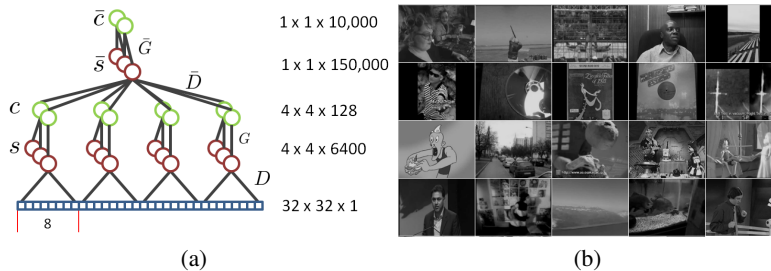

| | |
|---|---|
| | 1 x 1 x 10,000 |
| | 1 x 1 x 150,000 |
| | 4 x 4 x 128 |
| | 4 x 4 x 6400 |
| | 32 x 32 x 1 |

(a)           (b)

Figure 2: (a) Cross-section of network architecture used for experiments. Full layer sizes are shown at right. (b) Randomly selected 128-by-96 images from our dataset.

Our first layer of simple cell features are locally connected to 16 non-overlapping 8-by-8 pixel patches within the 32-by-32 pixel image. These features are trained by building a dataset of 8-by-8 patches and passing them to our simple cell learning procedure to train 6400 first-layer filters $D \in \Re^{64 \times 6400}$. We apply our complex cell learning procedure to this bank of filters to find 128 pooling groups $G_1, G_2, \ldots, G_{128}$. Using these results, we can extract our simple cell and complex cell features from each 8-by-8 pixel subpatch of the 32-by-32 image. Specifically, the linear filters $D$ are used to extract the first layer simple cell responses $s_i^{(p)} = g(D^{(i)\top} x^{(p)})$ where $x^{(p)}, p = 1, .., 16$ are the 16 subpatches of the 32-by-32 image. We then compute the complex cell feature responses $c_j^{(p)} = \max_{k \in G_j} s_k^{(p)}$ for each patch.

Once complete, we have an array of 128-by-4-by-4 = 2048 complex cell responses $c$ representing each 32-by-32 image. These responses are then used to form a new dataset from which to learn a second layer of simple cells with K-means. In our experiments we train 150,000 second layer simple cells. We denote the second layer of learned filters as $\bar{D}$, and the second layer simple cell responses as $\bar{s} = \bar{D}^\top c$. Applying again our complex cell learning procedure to $\bar{D}$, we obtain pooling groups $\bar{G}$, and complex cells $\bar{c}$ defined analogously.

## 3   Experiments

As described above, we ran our algorithm on patches harvested from YouTube thumbnails downloaded from the web. Specifically, we downloaded the thumbnails for over 1.4 million YouTube videos[4], some of which are shown in Figure 2b. These images were downsampled to 128-by-96 pixels and converted to grayscale. We cropped 57 million randomly selected 32-by-32 pixel patches from these images to form our unlabeled training set. No supervision was used—thus most patches contain partial views of objects or clutter at differing scales. We ran our algorithm on these images using a cluster of 30 machines over 3 days—virtually all of the time spent training the 150,000 second-layer features.[5] We will now visualize these features and check whether any of them have learned to identify an object class.

### 3.1   Low-Level Simple and Complex Cell Visualizations

We visualize the learned low-level filters $D$ and pooling groups $G$ to verify that they are, in fact, similar to those learned by other well-known algorithms. It is already known that our K-means-based algorithm learns simple-cell-like filters (e.g., edge-like features, as well as spots, curves) as shown in Figure 3a.

To visualize the learned complex cells we inspect the simple cell filters that belong to each of the pooling groups. The filters for several pooling groups are shown in Figure 3b. As expected, the filters cover a spectrum of similar image structures. Though many pairs of filters are extremely similar[6],

there are also other pairs that differ significantly yet are included in the group due to the single-link clustering method. Note that some of our groups are composed of similar edges at differing locations, and thus appear to have learned translation invariance as expected.

## 3.2 Higher-Level Simple and Complex Cells

Finally, we inspect the learned higher layer simple cell and complex cell features, $\bar{s}$ and $\bar{c}$, particularly to see whether any of them are selective for an object class. The most commonly occurring object in these video thumbnails is human faces (even though we estimate that much less than 0.1% of patches contain a well-framed face). Thus we search through our learned features for cells that are selective for human faces at varying locations and scales. To locate such features we use a dataset of labeled images: several hundred thousand non-face images as well as tens of thousands of known face images from the "Labeled Faces in the Wild" (LFW) dataset [4].[7]

To test whether any of the $\bar{s}$ simple cell features are selective for faces, we use each feature by itself as a "detector" on the labeled dataset: we compute the area under the precision-recall curve (AUC) obtained when each feature's response $\bar{s}_i$ is used as a simple classifier. Indeed, it turns out that there *are* a handful of high-level features that tend to be good detectors for faces. The precision-recall curves for the best 5 detectors are shown in Figure 3c (top curves); the best of these achieves 86% AUC. We visualize 16 of the simple cell features identified by this procedure[8] in Figure 4a along with a sampling of the image patches that activate the first of these cells strongly. There it can be seen that these simple cells are selective for faces located at particular locations and scales. Within each group the faces differ slightly due to the learned invariance provided by the complex cells in the lower layer (and thus the mean of each group of images is blurry).

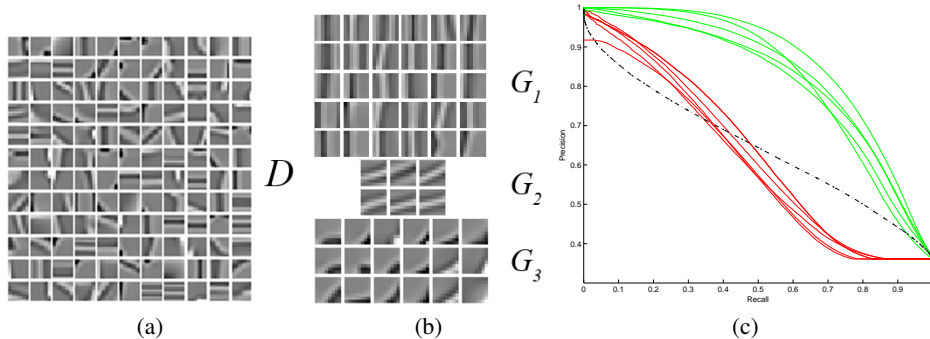

|   (a)   |   (b)   |   (c)   |

Figure 3: (a) First layer simple cell filters learned by K-means. (b) Sets of simple cell filters belonging to three pooling groups learned by our complex cell training algorithm. (c) Precision-Recall curves showing selectivity for human faces of 5 low-level simple cells trained from a full 32-by-32 patch (red curves, bottom) versus 5 higher-level simple cells (green curves, top). Performance of the best linear filter found by SVM from labeled data is also shown (black dotted curve, middle).

It may appear that this result could be obtained by applying our simple cell learning procedure directly to full 32-by-32 images without any attempts at incorporating local invariance. That is, rather than training $D$ (the first-layer filters) from 8-by-8 patches, we could try to train $D$ directly from the 32-by-32 images. This turns out not to be successful. The lower curves in Figure 3c are the precision-recall curves for the best 5 simple cells found in this way. Clearly the higher-level features are dramatically better detectors than simple cells built directly from pixels[9] (only 64% AUC).

| | Best 32-by-32 simple cell | Best in $\bar{s}$ | Best in $\bar{c}$ | Supervised Linear SVM |
|---|---|---|---|---|
| AUC | 64% | 86% | 80% | 77% |

Table 1: Area under PR curve for different cells on our face detection validation set. Only the SVM uses labeled data.

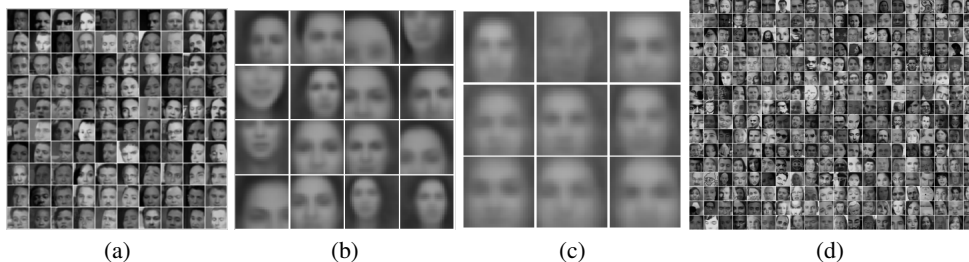

|     (a)     |     (b)     |     (c)     |     (d)     |

Figure 4: Visualizations. (a) A collection of patches from our unlabeled dataset that maximally activate one of the high-level simple cells from $\bar{s}$. (b) The mean of the top stimuli for a handful of face-selective cells in $\bar{s}$. (c) Visualization of the face-selective cells that belong to one of the complex cells in $\bar{c}$ discovered by the single-link clustering algorithm applied to $\bar{D}$. (d) A collection of unlabeled patches that elicit a strong response from the complex cell visualized in (c) — virtually all are faces, at a variety of scales and positions. Compare to (a).

As a second control experiment we train a linear SVM from half of the labeled data using only pixels as input (contrast-normalized and whitened). The PR curve for this linear classifier is shown in Figure 3c as a black dotted line. There we see that the supervised linear classifier is significantly better (77% AUC) than the 32-by-32 linear simple cells. On the other hand, it does not perform as well as the higher level simple cells learned by our system even though it is likely the best possible linear detector.

Finally, we inspect the higher-level complex cells learned by the applying the same agglomerative clustering procedure to the higher-level simple cell filters. Due to the invariance introduced at the lower layers, two simple cells that detect faces at slightly different locations or scales will often have very similar filter weights and thus we expect our algorithm to find and combine these simple cells into higher-level invariant features cells.

To visualize our higher-level complex cell features $\bar{c}$, we can simply look at visualizations for all of the simple cells in each of the groups $\bar{G}$. These visualizations show us the set of patches that strongly activate each simple cell, and hence also activate the complex cell. The results of such a visualization for one group that was found to contain only face-selective cells is shown in Figure 4c. There it can be seen that this single "complex cell" selects for faces at multiple positions and scales. A sampling of image patches collected from the unlabeled data that strongly activate the corresponding complex cell are shown in Figure 4d. We see that the complex cell detects many faces but at a much wider variety of positions and scales compared to the simple cells, demonstrating that even "higher level" invariances are being captured, including scale invariance. Benchmarked on our labeled set, this complex cell achieves 80.0% AUC—somewhat worse than the very best simple cells, but still in the top 10 performing cells in the entire network. Interestingly, the qualitative results in Figure 4d are excellent, and we believe these images represent an even greater range of variations than those in the labeled set. Thus the 80% AUC number may somewhat under-rate the quality of these features.

These results suggest that the basic notions of invariance and selectivity that underpin popular feature learning algorithms may be sufficient to discover the kinds of high-level features that we desire, possibly including whole object classes robust to local and global variations. Indeed, using simple implementations of selective and invariant features closely related to existing algorithms, we have found that is possible to build features with high selectivity for a coherent, commonly occurring object class. Though human faces occur only very rarely in our very large dataset, it is clear that the complex cell visualized Figure 4d is adept at spotting them amongst tens of millions of images. The enabler for these results is the scalability of the algorithms we have employed, suggesting that other systems can likely achieve similar results to the ones shown here if their computational limitations are overcome.

# 4  Related Work

The method that we have proposed has close connections to a wide array of prior work. For instance, the basic notions of selectivity and invariance that drive our system can be identified in many other algorithms: Group sparse coding methods [3] and Topographic ICA [6, 7] build invariances by pooling simple cells that lie in an invariant subspace, identified by strong scale correlations between cell responses. The advantage of this criterion is that it can determine which features to pool together even when the simple cell filters are orthogonal (where they would be too far apart for our algorithm to recognize their relationship). Our results suggest that while this type of invariance is very useful, there exist simple ways of achieving a similar effect.

Our approach is also connected with methods that attempt to model the geometric (e.g., manifold) structure of the input space. For instance, Contractive Auto-Encoders [16, 15], Local Coordinate Coding [20], and Locality-constrained Linear Coding [19] learn sparse linear filters while attempting to model the manifold structure staked out by these filters (sometimes termed "anchor points"). One interpretation of our method, suggested by Figure 1b, is that with extremely overcomplete dictionaries it is possible to use trivial distance calculations to identify neighboring points on the manifold. This in turn allows us to construct features invariant to shifts along the manifold with little effort. [1] use similar intuitions to propose a clustering method similar to our approach.

One of our key results, the unsupervised discovery of features selective for human faces is fairly unique (though seen recently in the extremely large system of [11]). Results of this kind have appeared previously in restricted settings. For instance, [13] trained Deep Belief Network models that decomposed object classes like faces and cars into parts using a probabilistic max-pooling to gain translation invariance. Similarly, [21] has shown results of a similar flavor on the Caltech recognition datasets. [22] showed that a probabilistic model (with some hand-coded geometric knowledge) can recover clusters containing 20 known object class silhouettes from outlines in the LabelMe dataset. Other authors have shown the ability to discover detailed manifold structure (e.g., as seen in the results of embedding algorithms [18, 17]) when trained in similarly restricted settings. The structure that these methods discover, however, is far more apparent when we are using labeled, *tightly cropped* images. Even if we do not use the labels themselves the labeled examples are, by construction, highly clustered: faces will be separated from other objects because there are no partial faces or random clutter. In our dataset, no supervision is used except to probe the representation post hoc.

Finally, we note the recent, extensive findings of Le et al. [11]. In that work an extremely large 9-layer neural network based on a TICA-like learning algorithm [10, 6] is also capable of identifying a wide variety of object classes (including cats and upper-bodies of people) seen in YouTube videos. Our results complement this work in several key ways. First, by training on smaller randomly cropped patches, we show that object-selectivity may still be obtained even when objects are almost never framed properly within the image—ruling out this bias as the source of object-selectivity. Second, we have shown that the key concepts (sparse selective filters and invariant-subspace pooling) used in their system can also be implemented in a different way using scalable clustering algorithms, allowing us to achieve results reminiscent of theirs using a vastly smaller amount of computing power. (We used 240 cores, while their large-scale system is composed of 16,000 cores.) In combination, these results point strongly to the conclusion that almost any highly scalable implementation of existing feature-learning concepts is enough to discover these sophisticated high-level representations.

# 5  Conclusions

In this paper we have presented a feature learning system composed of two highly scalable but otherwise very simple learning algorithms: K-means clustering to find sparse linear filters ("simple cells") and agglomerative clustering to stitch simple cells together into invariant features ("complex cells"). We showed that these two components are, in fact, capable of learning complicated high-level representations in large scale experiments on unlabeled images pulled from YouTube. Specifically, we found that higher level simple cells could learn to detect human faces without any supervision at all, and that our complex-cell learning procedure combined these into even higher-level invariances. These results indicate that we are apparently equipped with many of the key principles needed to achieve such results and that a critical remaining puzzle is how to scale up our algorithms to the sizes needed to capture more object classes and even more sophisticated invariances.

## Footnotes

[1]This allows us to train roughly half as many simple cell features for the first layer.

[2]Since the first layer uses $g(a) = |a|$, we actually use $d(k, l) = \min\{||D^{(k)} - D^{(l)}||_2, ||D^{(k)} + D^{(l)}||_2\}$ to account for $-D^{(l)}$ and $+D^{(l)}$ being essentially the same feature.

[3]We use $\tau = 0.3$ for the first layer of complex cells and $\tau = 1.0$ for the second layer. These were chosen by examining the typical distance between a filter $D^{(k)}$ and its nearest neighbor. We use $\Delta = 1.5 > \sqrt{2}$ so that a complex cell group may include orthogonal filters but cannot grow without limit.

[4]We cannot select videos at random, so we query videos under each YouTube category ("Pets & Animals", "Science & Technology", etc.) along with a date (e.g., "January 2001").

[5]Though this is a fairly long run, we note that 1 iteration of K-means is cheaper than a single batch gradient step for most other methods able to learn high-level invariant features. We expect that these experiments would be impossible to perform in a reasonable amount of time on our cluster with another algorithm.

[6]Some filters have reversed polarity due to our use of absolute-value rectification during training of the first layer.

[7]Our positive face samples include the entire set of labeled faces, plus randomly scaled and translated copies.

[8]We visualize the higher-level features by averaging together the 100 unlabeled images from our YouTube dataset that elicit the strongest activation.

[9]These simple cells were trained by applying K-means to normalized, whitened 32-by-32 pixel patches from a smaller unlabeled set known to have a higher concentration of faces. Due to this, a handful of centroids look roughly like face exemplars and act as simple "template matchers". When trained on the full dataset (which contains far fewer faces), K-means learns only edge and arc features which perform much worse (about 45% AUC).

# References

[1] Y. Boureau, N. L. Roux, F. Bach, J. Ponce, and Y. LeCun. Ask the locals: multi-way local pooling for image recognition. In *13th International Conference on Computer Vision*, pages 2651–2658, 2011.

[2] A. Coates and A. Y. Ng. The importance of encoding versus training with sparse coding and vector quantization. In *International Conference on Machine Learning*, pages 921–928, 2011.

[3] P. Garrigues and B. Olshausen. Group sparse coding with a laplacian scale mixture prior. In *Advances in Neural Information Processing Systems 23*, pages 676–684, 2010.

[4] G. B. Huang, M. Ramesh, T. Berg, and E. Learned-Miller. Labeled faces in the wild: A database for studying face recognition in unconstrained environments. Technical Report 07-49, University of Massachusetts, Amherst, October 2007.

[5] A. Hyvärinen and P. Hoyer. Emergence of phase-and shift-invariant features by decomposition of natural images into independent feature subspaces. *Neural Computation*, 12(7):1705–1720, 2000.

[6] A. Hyvärinen, P. Hoyer, and M. Inki. Topographic independent component analysis. *Neural Computation*, 13(7):1527–1558, 2001.

[7] A. Hyvärinen, J. Hurri, and P. Hoyer. *Natural Image Statistics*. Springer-Verlag, 2009.

[8] T. Kohonen. Emergence of invariant-feature detectors in self-organization. In M. Palaniswami et al., editor, *Computational Intelligence, A Dynamic System Perspective*, pages 17–31. IEEE Press, New York, 1995.

[9] A. Krizhevsky. Learning multiple layers of features from Tiny Images. Master's thesis, Dept. of Comp. Sci., University of Toronto, 2009.

[10] Q. Le, A. Karpenko, J. Ngiam, and A. Ng. ICA with reconstruction cost for efficient overcomplete feature learning. In *Advances in Neural Information Processing Systems*, 2011.

[11] Q. Le, M. Ranzato, R. Monga, M. Devin, K. Chen, G. Corrado, J. Dean, and A. Ng. Building high-level features using large scale unsupervised learning. In *International Conference on Machine Learning*, 2012.

[12] Y. LeCun, B. Boser, J. S. Denker, D. Henderson, R. E. Howard, W. Hubbard, and L. D. Jackel. Backpropagation applied to handwritten zip code recognition. *Neural Computation*, 1:541–551, 1989.

[13] H. Lee, R. Grosse, R. Ranganath, and A. Y. Ng. Convolutional deep belief networks for scalable unsupervised learning of hierarchical representations. In *International Conference on Machine Learning*, pages 609–616, 2009.

[14] M. Riesenhuber and T. Poggio. Hierarchical models of object recognition in cortex. *Nature neuroscience*, 2, 1999.

[15] S. Rifai, Y. Dauphin, P. Vincent, Y. Bengio, and X. Muller. The manifold tangent classifier. In *Advances in Neural Information Processing*, 2011.

[16] S. Rifai, P. Vincent, X. Muller, X. Glorot, and Y. Bengio. Contractive auto-encoders: Explicit invariance during feature extraction. In *International Conference on Machine Learning*, 2011.

[17] S. Roweis and L. Saul. Nonlinear dimensionality reduction by locally linear embedding. *Science*, 290(5500):2323—2326, December 2000.

[18] L. van der Maaten and G. Hinton. Visualizing high-dimensional data using t-SNE. *Journal of Machine Learning Research*, 9:2579—2605, November 2008.

[19] J. Wang, J. Yang, K. Yu, F. Lv, T. Huang, and Y. Gong. Locality-constrained linear coding for image classification. In *Computer Vision and Pattern Recognition*, pages 3360–3367, 2010.

[20] K. Yu, T. Zhang, and Y. Gong. Nonlinear learning using local coordinate coding. In *Advances in Neural Information Processing Systems 22*, pages 2223–2231, 2009.

[21] M. D. Zeiler, G. W. Taylor, and R. Fergus. Adaptive deconvolutional networks for mid and high level feature learning. In *International Conference on Computer Vision*, 2011.

[22] L. Zhu, Y. Chen, A. Torralba, W. Freeman, and A. Yuille. Part and Appearance Sharing: Recursive Compositional Models for Multi-View Multi-Object Detection. In *Computer Vision and Pattern Recognition*, 2010.

